# Nonparametric Bayesian Models for Unsupervised Event Coreference Resolution

**Cosmin Adrian Bejan**[1]**, Matthew Titsworth**[2]**, Andrew Hickl**[2]**, & Sanda Harabagiu**[1]
[1] Human Language Technology Research Institute, University of Texas at Dallas
[2] Language Computer Corporation, Richardson, Texas
`ady@hlt.utdallas.edu`

## Abstract

We present a sequence of unsupervised, nonparametric Bayesian models for clustering complex linguistic objects. In this approach, we consider a potentially infinite number of features and categorical outcomes. We evaluated these models for the task of within- and cross-document event coreference on two corpora. All the models we investigated show significant improvements when compared against an existing baseline for this task.

## 1   Introduction

In Natural Language Processing (NLP), the task of event coreference has numerous applications, including question answering, multi-document summarization, and information extraction. Two event mentions are *coreferential* if they share the same participants and spatio-temporal groundings. Moreover, two event mentions are *identical* if they have the same causes and effects. For example, the three documents listed in Table 1 contains four mentions of identical events but only the *arrested*, *apprehended*, and *arrest* mentions from the documents 1 and 2 are coreferential. These definitions were used in the tasks of Topic Detection and Tracking (TDT), as reported in [24].

Previous approaches to event coreference resolution [3] used the same lexeme or synonymy of the verb describing the event to decide coreference. Event coreference was also tried by using the semantic types of an ontology [17]. However, the features used by these approaches are hard to select and require the design of domain specific constraints. To address this problems, we have explored a sequence of unsupervised, nonparametric Bayesian models that are used to probabilistically infer coreference clusters of event mentions from a collection of unlabeled documents. Our approach is motivated by the recent success of unsupervised approaches for entity coreference resolution [16, 22, 25] and by the advantages of using a large amount of data at no cost.

One model was inspired by the fully generative Bayesian model proposed by Haghighi and Klein [16] (henceforth, H&K). However, to employ the H&K's model for tasks that require clustering objects with rich linguistic features (such as event coreference resolution), or to extend this model in order to enclose additional observable properties is a challenging task [22, 25]. In order to counter this limitation, we make a conditional independence assumption between the observable features and propose a generalized framework (Section 3) that is able to easily incorporate new features.

During the process of learning the model described in Section 3, it was observed that a large amount of time was required to incorporate and tune new features. This lead us to the challenge of creating a framework which considers an unbounded number of features where the most relevant are selected automatically. To accomplish this new goal, we propose two novel approaches (Section 4). The first incorporates a *Markov Indian Buffet Process* (mIBP) [30] into a *Hierarchical Dirichlet Process* (HDP) [28]. The second uses an *Infinite Hidden Markov Model* (iHMM) [5] coupled to an *Infinite Factorial Hidden Markov Model* (iFHMM) [30].

In this paper, we focus on event coreference resolution, though adaptations for event identity resolution can be easily made. We evaluated the models on the ACE 2005 event corpus [18] and on a new annotated corpus encoding within- and cross-document event coreference information (Section 5).

| |
|---|
| **Document 1:** *San Diego Chargers receiver Vincent Jackson was **arrested** on suspicion of drunk driving on Tuesday morning, five days before a key NFL playoff game.* <br> *⋯* <br> *Police **apprehended** Jackson in San Diego at 2:30 a.m. and booked him for the misdemeanour before his release.* |
| **Document 2:** *Despite his **arrest** on suspicion of driving under the influence yesterday, Chargers receiver Vincent Jackson will play in Sunday's AFC divisional playoff game at Pittsburgh.* |
| **Document 3:** *In another anti-piracy operation, Navy warship on Saturday repulsed an attack on a merchant vessel in the Gulf of Aden and **nabbed** 23 Somali and Yemeni sea brigands.* |

Table 1: Examples of coreferential and identical events.

## 2 Event Coreference Resolution

Models for solving event coreference and event identity can lead to the generation of ad-hoc event hierarchies from text. A sample of a hierarchy capturing corefering and identical events, including those from the example presented in Section 1, is illustrated in Figure 1.

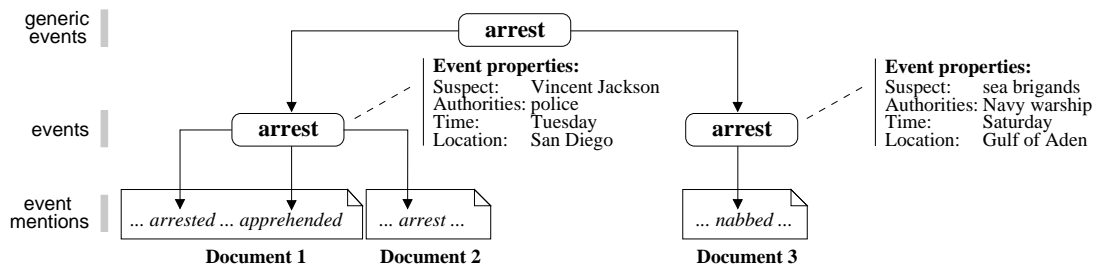

Figure 1: A portion of the event hierarchy.

First, we introduce some basic notation.[1] Next, to cluster the mentions that share common event properties (as shown in Figure 1), we briefly describe the linguistic features of event mentions.

### 2.1 Notation

As input for our models, we consider a collection of $I$ documents, each document $i$ having $J_i$ event mentions. Each event mention is characterized by $L$ *feature types*, FT, and each feature type is represented by a finite number of *feature values*, $fv$. Therefore, we can represent the observable properties of an event mention, $em$, as a vector of pairs $\langle (\text{FT}_1 : fv_{1i}), \dots, (\text{FT}_L : fv_{Li}) \rangle$, where each feature value index $i$ ranges in the feature value space associated with a feature type.

### 2.2 Linguistic Features

We consider the following set of features associated to an event mention:[2]

**Lexical Features (LF)** To capture the lexical context of an event mention, we extract the following features: the head word of the mention (HW), the lemma of the HW (HL), lemmas of left and right words of the mention (LHL,RHL), and lemmas of left and right mentions (LHE,RHE).

**Class Features (CF)** These features aim to classify mentions into several types of classes: the mention HW's part-of-speech (POS), the word class of the HW (HWC), which can take one of the following values ⟨*verb*, *noun*, *adjective*, *other*⟩, and the event class of the mention (EC). To extract the event class associated to every event mention, we employed the event identifier described in [6].

**WordNet Features (WF)** We build three types of clusters over all the words from WordNet [9] and use them as features for the mention HW. First cluster type associates an unique $id$ to each (word:HWC) pair (WNW). The second cluster type uses the transitive closure of the *synonymous* relations to group words from WordNet (WNS). Finally, the third cluster type considers as grouping criteria the category from WordNet lexicographer's files that is associated to each word (WNL). For cases when a new word does not belong to any of these WordNet clusters, we create a new cluster with a new $id$ for each of the three cluster types.

**Semantic Features (SF)** To extract features that characterize participants and properties of event mentions, we use s semantic parser [8] trained on PropBank(PB) [23] and FrameNet(FN) [4] corpora. (For instance, for the *apprehended* mention from our example, *Jackson* is the feature value

for A0 PB argument[3] and the SUSPECT frame element (FEA0) of the ARREST frame.) Another semantic feature is the semantic frame (FR) that is evoked by an event mention. (For instance, all the emphasized mentions from our example evoke the ARREST frame from FN.)

**Feature Combinations (FC)** We also explore various combinations of features presented above. Examples include HW+POS, HL+FR, FE+A1, etc.

## 3 Finite Feature Models

In this section, we present a sequence of HDP mixture models for solving event coreference. For this type of approach, a *Dirichlet Process* (DP) [10] is associated with each document, and each mixture component, which in our case corresponds to an event, is shared across documents. To describe these models, we consider $\mathbf{Z}$ the set of indicator random variables for indices of events, $\phi_z$ the set of parameters associated to an event $z$, $\phi$ a notation for all model parameters, and $\mathbf{X}$ a notation for all random variables that represent observable features.

Given a document collection annotated with event mentions, the goal is to find the best assignment of event indices, $\mathbf{Z}^*$, which maximize the posterior probability $P(\mathbf{Z} \mid \mathbf{X})$. In a Bayesian approach, this probability is computed by integrating out all model parameters:

$$P(\mathbf{Z}|\mathbf{X}) = \int P(\mathbf{Z}, \phi|\mathbf{X})d\phi = \int P(\mathbf{Z}|\mathbf{X}, \phi)P(\phi|\mathbf{X})d\phi$$

In order to describe our modifications, we first revisit a basic model from the set of models described in H&K's paper.

### 3.1 The One Feature Model

The one feature model, $\text{HDP}_{1f}$, constitutes the simplest representation of an HDP model. In this model, which is depicted graphically in Figure 2(a), the observable components are characterized by only one feature. The distribution over events associated to each document $\beta$ is generated by a Dirichlet process with a concentration parameter $\alpha > 0$. Since this setting enables a clustering of event mentions at the document level, it is desirable that events are shared across documents and the number of events $K$ is inferred from data. To ensure this flexibility, a global nonparametric DP prior with a hyperparameter $\gamma$ and a global base measure $H$ can be considered for $\beta$ [28]. The global distribution drawn from this DP prior, denoted as $\beta_0$ in Figure 2(a), encodes the event mixing weights. Thus, same global events are used for each document, but each event has a document specific distribution $\beta_i$ that is drawn from a DP prior centered on $\beta_0$.

To infer the true posterior probability of $P(\mathbf{Z}|\mathbf{X})$, we follow [28] in using a Gibbs sampling algorithm [12] based on the direct assignment sampling scheme. In this sampling scheme, the $\beta$ and $\phi$ parameters are integrated out analytically. The formula for sampling an event index for mention $j$ from document $i$, $Z_{i,j}$, is given by:[4]

$$P(Z_{i,j} \mid \mathbf{Z}^{-i,j}, \mathbf{HL}) \propto P(Z_{i,j} \mid \mathbf{Z}^{-i,j})P(HL_{i,j} \mid \mathbf{Z}, \mathbf{HL}^{-i,j})$$

where $HL_{i,j}$ is the head lemma of the event mention $j$ from the document $i$.

First, in the generative process of an event mention, an event index $z$ is sampled by using a mechanism that facilitates sampling from a prior for infinite mixture models called the Chinese Restaurant Franchise (CRF) representation [28]:

$$P(Z_{i,j} = z \mid \mathbf{Z}^{-i,j}, \beta_0) \propto \begin{cases} \alpha\beta_0^u, & \text{if } z = z_{new} \\ n_z + \alpha\beta_0^z, & \text{otherwise} \end{cases}$$

Here, $n_z$ is the number of event mentions with the event index $z$, $z_{new}$ is a new event index not used already in $\mathbf{Z}^{-i,j}$, $\beta_0^z$ are the global mixing proportions associated to the $K$ events, and $\beta_0^u$ is the weight for the unknown mixture component.

Then, to generate the mention head lemma (in this model, $\mathbf{X} = \langle \mathbf{HL} \rangle$), the event $z$ is associated with a multinomial emission distribution over the HL feature values having the parameters $\phi = \langle \phi_Z^{hl} \rangle$. We assume that this emission distribution is drawn from a symmetric Dirichlet distribution with concentration $\lambda_{HL}$:

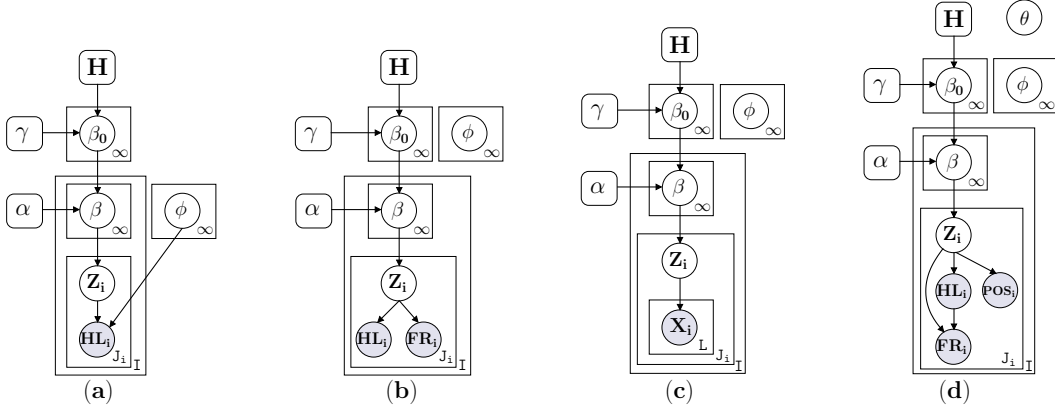

Figure 2: Graphical representation of four HDP models. Each node corresponds to a random variable. In particular, shaded nodes denotes observable variables. Each rectangle captures the replication of the structure it contains. The number of replications is indicated in the bottom-right corner of the rectangle. The model depicted in (a) is an HDP model using one feature; the model in (b) employs HL and FR features; (c) illustrates a flat representation of a limited number of features in a generalized framework (henceforth, $\text{HDP}_{flat}$); and (d) captures a simple example of structured network topology of three feature variables (henceforth, $\text{HDP}_{struct}$). The dependencies involving parameters $\phi$ and $\theta$ in models (b), (c), and (d) are omitted for clarity.

$$P(HL_{i,j} = hl \mid \mathbf{Z}, \mathbf{HL}^{-i,j}) \propto n_{hl,z} + \lambda_{HL}$$

where $HL_{i,j}$ is the head lemma of mention $j$ from document $i$, and $n_{hl,z}$ is the number of times the feature value $hl$ has been associated with the event index $z$ in $(\mathbf{Z}, \mathbf{HL}^{-i,j})$. We also apply the Lidstone's smoothing method to this distribution.

### 3.2 Adding More Features

A model in which observable components are represented only by one feature has the tendency to cluster these components based on their feature value. To address this limitation, H&K proposed a more complex model that is strictly customized for entity coreference resolution. On the other hand, event coreference involves clustering complex objects characterized by richer features than entity coreference (or topic detection), and therefore it is desirable to extend the $\text{HDP}_{1f}$ model with a generalized model where additional features can be easily incorporated.

To facilitate this extension, we assume that feature variables are conditionally independent given $\mathbf{Z}$. This assumption considerably reduces the complexity of computing $P(\mathbf{Z} \mid \mathbf{X})$. For example, if we want to incorporate another feature (e.g., $FR$) in the previous model, the formula becomes:

$$P(Z_{i,j} \mid \mathbf{HL}, \mathbf{FR}) \propto P(Z_{i,j})P(HL_{i,j}, FR_{i,j} \mid \mathbf{Z}) = P(Z_{i,j})P(HL_{i,j} \mid \mathbf{Z})P(FR_{i,j} \mid \mathbf{Z})$$

In this formula, we omit the conditioning components of $\mathbf{Z}$, $\mathbf{HL}$, and $\mathbf{FR}$ for clarity. The graphical representation corresponding to this model is illustrated in Figure 2(b). In general, if $\mathbf{X}$ consists of $L$ feature variables, the inference formula for the Gibbs sampler is defined as:

$$P(Z_{i,j} \mid \mathbf{X}) \propto P(Z_{i,j}) \prod_{FT \in \mathbf{X}} P(FT_{i,j} \mid \mathbf{Z})$$

The graphical model for this general setting is depicted in Figure 2(c). Drawing an analogy, the graphical representation involving feature variables and $\mathbf{Z}$ variables resembles the graphical representation of a Naive Bayes classifier.

When dependencies between feature variables exist (e.g., in our case, frame elements are dependent of the semantic frames that define them, and frames are dependent of the words that evoke them), various global distributions are involved for computing $P(\mathbf{Z} \mid \mathbf{X})$. For instance, for the model depicted in Figure 2(d) the posterior probability is given by:

$$P(Z_{i,j})P(FR_{i,j} \mid HL_{i,j}, \boldsymbol{\theta}) \prod_{FT \in \mathbf{X}} P(FT_{i,j} \mid \mathbf{Z})$$

In this model, $P(FR_{i,j} \mid HL_{i,j}, \boldsymbol{\theta})$ is a global distribution parameterized by $\boldsymbol{\theta}$, and the feature variables considered are $\mathbf{X} = \langle \mathbf{HL}, \mathbf{POS}, \mathbf{FR} \rangle$.

For all these extended models, we compute the prior and likelihood factors as described in the one feature model. Also, following H&K, in the inference mechanism we assign soft counts for missing features (e.g., unspecified PB argument).

## 4 Unbounded Feature Models

First, we present a generative model called the *Markov Indian Buffet Process* (mIBP) that provides a mechanism in which each object can be represented by a sparse subset of a potentially unbounded set of latent features [15, 14, 30].[5] Then, to overcome the limitations regarding the number of mixture components and the number of features associated with objects, we combine this mechanism with an HDP model to form an mIBP-HDP hybrid. Finally, to account for temporal dependencies, we employ an mIBP extension, called the *Infinite Factorial Hidden Markov Model* (iFHMM) [30], in combination with an *Infinite Hidden Markov Model* (iHMM) to form the iFHMM-iHMM model.

### 4.1 The Markov Indian Buffet Process

As described in [30], the mIBP defines a distribution over an unbounded set of binary Markov chains, where each chain can be associated to a binary latent feature that evolves over time according to Markov dynamics. Specifically, if we denote by $M$ the total number of feature chains and by $T$ the number of observable components (event mentions), the mIBP defines a probability distribution over a binary matrix $\mathbf{F}$ with $T$ rows, which correspond to observations, and an unbounded number of columns ($M \to \infty$), which correspond to features. An observation $y_t$ contains a subset from the unbounded set of features $\{f^1, f^2, \ldots, f^M\}$ that is represented in the matrix by a binary vector $\mathbf{F_t} = \langle F_t^1, F_t^2, \ldots, F_t^M \rangle$, where $F_t^i = 1$ indicates that $f^i$ is associated to $y_t$.

Therefore, $\mathbf{F}$ decomposes the observations and represents them as feature factors, which can then be associated to hidden variables in an iFHMM as depicted in Figure 3(a). The transition matrix of a binary Markov chain associated to a feature $f^m$ is defined as

$$\mathbf{W}^{(m)} = \begin{pmatrix} 1 - a_m & a_m \\ 1 - b_m & b_m \end{pmatrix}$$

where $\mathbf{W}_{ij}^{(m)} = P(F_{t+1}^m = j \mid F_t^m = i)$, the parameters $a_m \sim \mathsf{Beta}(\alpha'/M, 1)$ and $b_m \sim \mathsf{Beta}(\gamma', \delta')$, and the initial state $F_0^m = 0$. In the generative process, the hidden variable of feature $f^m$ for an object $y_t$, $F_t^m \sim \mathsf{Bernoulli}(a_m^{1-F_{t-1}^m} b_m^{F_{t-1}^m})$.

To compute the probability of the feature matrix $\mathbf{F}$[6], in which the parameters $\mathbf{a}$ and $\mathbf{b}$ are integrated out analytically, we use the counting variables $c_m^{00}$, $c_m^{01}$, $c_m^{10}$, and $c_m^{11}$ to record the $0 \to 0$, $0 \to 1$, $1 \to 0$, and $1 \to 1$ transitions $f^m$ has made in the binary chain $m$. The stochastic process that derives the probability distribution in terms of these variables is defined as follows. The first component samples a number of $\mathsf{Poisson}(\alpha')$ features. In general, depending on the value that was sampled in the previous step $(t - 1)$, a feature $f^m$ is sampled for the $t^{th}$ component according to the following probabilities:

$$P(F_t^m = 1 \mid F_{t-1}^m = 1) = \frac{c_m^{11} + \delta'}{\gamma' + \delta' + c_m^{10} + c_m^{11}}$$

$$P(F_t^m = 1 \mid F_{t-1}^m = 0) = \frac{c_m^{00}}{c_m^{00} + c_m^{01}}$$

The $t^{th}$ component then repeats the same mechanism for sampling the next features until it finishes the current number of sampled features $M$. After all features are sampled for the $t^{th}$ component, a number of $\mathsf{Poisson}(\alpha'/t)$ new features are assigned for this component and $M$ gets incremented accordingly.

### 4.2 The mIBP-HDP Model

One direct application of the mIBP is to integrate it into the HDP models proposed in Section 3. In this way, the new nonparametric extension will have the benefits of capturing uncertainty regarding the number of mixture components that are characterized by a potentially infinite number of features. Since one observable component is associated with an unbounded countable set of features, we have to provide a mechanism in which only a finite set of features will represent the component in the HDP inference process.

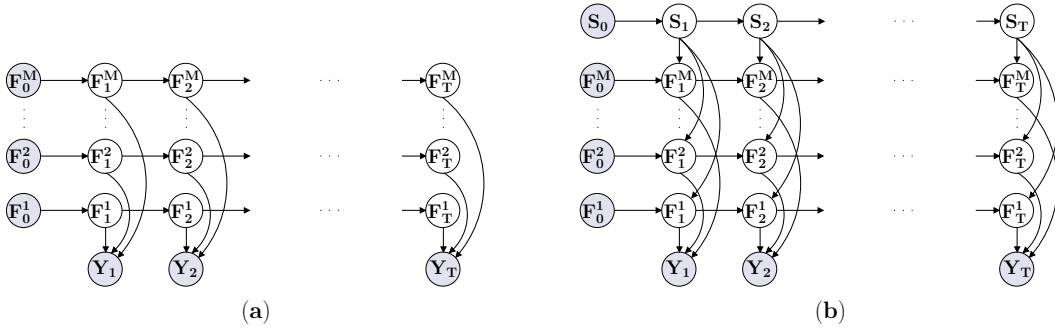

Figure 3: (a) The Infinite Factorial Hidden Markov Model. (b) The iFHMM-iHMM model. ($M \rightarrow \infty$)

The idea behind this mechanism is to use *slice sampling*[7] [21] in order to derive a finite set of features for $y_t$. Letting $q_m$ be the number of times feature $f^m$ was sampled in the mIBP, and $v_t$ an auxiliary variable for $y_t$ such that $v_t \sim \mathsf{Uniform}(1, \max\{q_m \mid F_t^m = 1\})$, we define the finite feature set $B_t$ for the observation $y_t$ as:

$$B_t = \{f^m \mid F_t^m = 1 \land q_m \geq v_t\}$$

The finiteness of this feature set is based on the observation that, in the generative process of the mIBP, only a finite set of features are sampled for a component. Another observation worth mentioning regarding the way this set is constructed is that only the most representative features of $y_t$ get selected in $B_t$.

### 4.3 The iFHMM-iHMM Model

The iFHMM is a nonparametric Bayesian factor model that extends the *Factorial Hidden Markov Model* (FHMM) [13] by letting the number of parallel Markov chains $M$ be learned from data. Although the iFHMM allows a more flexible representation of the latent structure, it can not be used as a framework where the number of clustering components $K$ is infinite. On the other hand, the iHMM represents a nonparametric extension of the Hidden Markov Model (HMM) [27] that allows performing inference on an infinite number of states $K$. In order to further increase the representational power for modeling discrete time series data, we propose a nonparametric extension that combines the best of the two models, and lets the parameters $M$ and $K$ be learned from data.

Each step in the new generative process, whose graphical representation is depicted in Figure 3(b), is performed in two phases: (i) the latent feature variables from the iFHMM framework are sampled using the mIBP mechanism; and (ii) the features sampled so far, which become observable during this second phase, are used in an adapted *beam sampling algorithm* [29] to infer the clustering components (or, in our case, latent events).

To describe the beam sampler for event coreference resolution, we introduce additional notation. We denote by $(s_1, \ldots, s_T)$ the sequence of hidden states corresponding to the sequence of event mentions $(y_1, \ldots, y_T)$, where each state $s_t$ belong to one of the $K$ events, $s_t \in \{1, \ldots, K\}$, and each mention $y_t$ is represented by a sequence of latent features $\langle F_t^1, F_t^2, \ldots, F_t^M \rangle$. One element of the transition probability $\boldsymbol{\pi}$ is defined as $\pi_{ij} = P(s_t = j \mid s_{t-1} = i)$ and a mention $y_t$ is generated according to a likelihood model $\mathcal{F}$ that is parameterized by a state-dependent parameter $\phi_{s_t}$ ($y_t \mid s_t \sim \mathcal{F}(\phi_{s_t})$). The observation parameters $\phi$ are iid drawn from a prior base distribution $H$.

The beam sampling algorithm combines the ideas of slice sampling and dynamic programming for an efficient sampling of state trajectories. Since in time series models the transition probabilities have independent priors [5], Van Gael and colleagues [29] also used the HDP mechanism to allow couplings across transitions. For sampling the whole hidden state trajectory $\mathbf{s}$, this algorithm employs a forward filtering-backward sampling technique.

In the forward step of our implementation, we sample the feature variables using the mIBP as described in Section 4.1, and the auxiliary variable $u_t \sim \mathsf{Uniform}(0, \pi_{s_{t-1}s_t})$ for each mention $y_t$. As explained in [29], the auxiliary variables $\mathbf{u}$ are used to filter only those trajectories $\mathbf{s}$ for which

$\pi_{s_{t-1}s_t} \geq u_t$ for all $t$. Also, in this step, we compute the probabilities $P(s_t \mid y_{1:t}, u_{1:t})$ for all $t$ as described in [29]:

$$P(s_t \mid y_{1:t}, u_{1:t}) \propto P(y_t \mid s_t) \sum_{s_{t-1}:u_t < \pi_{s_{t-1}s_t}} P(s_{t-1} \mid y_{1:t-1}, u_{1:t-1})$$

Here, the dependencies involving parameters $\pi$ and $\phi$ are omitted for clarity.

In the backward step, we first sample the event for the last state $s_T$ directly from $P(s_T \mid y_{1:T}, u_{1:T})$ and then, for all $t : T - 1, 1$, we sample each state $s_t$ given $s_{t+1}$ by using the formula $P(s_t \mid s_{t+1}, y_{1:T}, u_{1:T}) \propto P(s_t \mid y_{1:t}, u_{1:t}) P(s_{t+1} \mid s_t, u_{t+1})$.

To sample the emission distribution $\phi$ efficiently, and to ensure that each mention is characterized by a finite set of representative features, we set the base distribution $H$ to be conjugate with the data distribution $\mathcal{F}$ in a Dirichlet-multinomial model with the sufficient statistics of the multinomial distribution $(o_1, \dots, o_K)$ defined as:

$$o_k = \sum_{t=1}^{T} \sum_{f^m \in B_t} n_{mk}$$

where $n_{mk}$ counts how many times feature $f^m$ was sampled for event $k$, and $B_t$ stores a finite set of features for $y_t$ as it is defined in Section 4.2.

## 5 Evaluation

**Event Coreference Data** One corpus used for evaluation is ACE 2005 [18]. This corpus annotates within-document coreference information of specific types of events (such as *Conflict*, *Justice*, and *Life*). After an initial processing phase, we extracted from ACE 6553 event mentions and 4946 events. To increase the diversity of events and to evaluate the models for both within- and cross-document event coreference, we created the EventCorefBank corpus (ECB).[8] This new corpus contains 43 topics, 1744 event mentions, 1302 within-document events, and 339 cross-document events.

For a more realistic approach, we trained the models on all the event mentions from the two corpora and not only on the mentions manually annotated for event coreference (the true event mentions). In this regard, we ran the event identifier described in [6] on the ACE and ECB corpora, and extracted 45289 and 21175 system mentions respectively.

**The Experimental Setup** Table 2 lists the recall (R), precision (P), and F-score (F) of our experiments averaged over 5 runs of the generative models. Since there is no agreement on the best coreference resolution metric, we employed four metrics for our evaluation: the *link*-based MUC metric [31], the *mention*-based B$^3$ metric [2], the *entity*-based CEAF metric [19], and the pairwise F1 (PW) metric. In the evaluation process, we considered only the true mentions of the ACE test dataset and of the test sets of a 5-fold cross validation scheme on the ECB dataset. For evaluating the cross-document coreference annotations, we adopted the same approach as described in [3] by merging all the documents from the same topic into a meta-document and then scoring this document as performed for within-document evaluation. Also, for both corpora, we considered a set of 132 feature types, where each feature type consists on average of 3900 distinct feature values.

**The Baseline** A simple baseline for event coreference consists in grouping events by their event classes [1]. To extract event classes, we employed the event identifier described in [6]. Therefore, this baseline will categorize events into a small number of clusters, since the event identifier is trained to predict the five event classes annotated in TimeBank [26]. As it was already observed [20, 11], considering very few categories for coreference resolution tasks will result in overestimates of the MUC scorer. For instance, a baseline that groups all entity mentions into the same entity achieves the highest MUC score than any published system for the task of entity coreference. Similar behaviour of the MUC metric is observed for event coreference resolution. For example, for cross-document evaluation on ECB, a baseline that clusters all mentions into one event achieves 73.2% MUC F-score, while the baseline listed in Table 2 achieves 72.9% MUC F-score.

**HDP Extensions** Due to memory limitations, we evaluated the HDP$_{flat}$ and HDP$_{struct}$ models only on a restricted subset of manually selected feature types. In general, as shown in Table 2, the HDP$_{flat}$ model achieved the best performance results on the ACE test dataset, whereas the

| Model | MUC | | | B$^3$ | | | CEAF | | | PW | | |
|---|---|---|---|---|---|---|---|---|---|---|---|---|
| | R | P | F | R | P | F | R | P | F | R | P | F |
| ACE (within-document event coreference) | | | | | | | | | | | | |
| Baseline | 94.3 | 33.1 | 49.0 | 97.9 | 25.0 | 39.9 | 14.7 | 64.4 | 24.0 | 93.5 | 8.2 | 15.2 |
| HDP$_{1f}$ (HL) | 62.2 | 43.1 | 50.9 | 86.0 | 70.6 | 77.5 | 62.3 | 76.4 | 68.6 | 50.5 | 27.7 | 35.8 |
| HDP$_{flat}$ | 53.5 | 54.2 | 53.9 | 83.4 | 84.2 | **83.8** | 76.9 | 76.5 | **76.7** | 43.3 | 47.1 | **45.1** |
| HDP$_{struct}$ | 61.9 | 49.0 | **54.7** | 86.2 | 76.9 | 81.3 | 69.0 | 77.5 | 73.0 | 53.2 | 38.1 | 44.4 |
| mIBP-HDP | 48.7 | 41.9 | 45.1 | 81.7 | 76.4 | 79.0 | 68.8 | 73.8 | 71.2 | 37.4 | 28.9 | 32.6 |
| iFHMM-iHMM | 48.7 | 48.8 | 48.7 | 81.9 | 82.2 | 82.1 | 74.6 | 74.5 | 74.5 | 37.2 | 39.0 | 38.1 |
| ECB (within-document event coreference) | | | | | | | | | | | | |
| Baseline | 92.2 | 39.8 | 55.6 | 97.7 | 55.8 | 71.0 | 44.5 | 80.1 | 57.2 | 93.7 | 25.4 | 39.8 |
| HDP$_{1f}$ (HL) | 46.9 | 54.8 | 50.4 | 84.3 | 89.0 | 86.5 | 83.4 | 79.6 | 81.4 | 36.6 | 53.4 | 42.6 |
| HDP$_{flat}$ | 37.8 | 92.9 | 53.4 | 82.1 | 99.2 | 89.8 | 93.9 | 78.2 | 85.3 | 27.0 | 92.4 | 41.3 |
| HDP$_{struct}$ | 47.4 | 82.7 | **60.1** | 84.3 | 97.1 | **90.2** | 92.7 | 81.1 | **86.5** | 34.4 | 83.0 | **48.6** |
| mIBP-HDP | 38.2 | 68.8 | 48.9 | 82.1 | 95.3 | 88.2 | 90.3 | 78.5 | 84.0 | 26.5 | 67.9 | 37.7 |
| iFHMM-iHMM | 39.5 | 85.2 | 53.9 | 82.5 | 98.1 | 89.6 | 93.1 | 78.8 | 85.3 | 29.4 | 86.6 | 43.7 |
| ECB (cross-document event coreference) | | | | | | | | | | | | |
| Baseline | 90.5 | 61.1 | **72.9** | 93.8 | 49.6 | 64.9 | 36.6 | 72.7 | 48.7 | 90.7 | 28.6 | 43.3 |
| HDP$_{1f}$ (HL) | 47.7 | 70.5 | 56.8 | 67.0 | 86.2 | 75.3 | 76.2 | 57.1 | 65.2 | 34.9 | 58.9 | 43.5 |
| HDP$_{flat}$ | 44.4 | 95.3 | 60.5 | 65.0 | 98.7 | 78.3 | 86.9 | 56.0 | 68.0 | 29.2 | 95.1 | 44.4 |
| HDP$_{struct}$ | 51.9 | 89.5 | 65.7 | 69.3 | 95.8 | **80.4** | 86.2 | 60.1 | **70.8** | 37.5 | 85.6 | **52.1** |
| mIBP-HDP | 40.0 | 79.8 | 53.2 | 63.1 | 94.1 | 75.5 | 82.7 | 54.6 | 65.7 | 26.1 | 77.0 | 38.9 |
| iFHMM-iHMM | 48.4 | 89.0 | 62.7 | 67.0 | 96.4 | 79.0 | 85.5 | 58.0 | 69.1 | 33.3 | 88.3 | 48.2 |

Table 2: Evaluation results for within- and cross-document event coreference resolution.

HDP$_{struct}$ model, which also considers dependencies between feature types, proved to be more effective on the ECB dataset for both within- and cross-document event coreference evaluation. The set of feature types used to achieve these results consists of combinations of types from all feature categories described in Section 2.2. For the results of the HDP$_{struct}$ model listed in Table 2, we also explored the conditional dependencies between the HL, FR, and FEA types.

As can be observed from Table 2, the results of the HDP$_{flat}$ and HDP$_{struct}$ models show an F-score increase by 4-10% over the HDP$_{1f}$ model, and therefore prove that the HDP extensions provide a more flexible representation for clustering objects characterized by rich properties.

**mIBP-HDP** In spite of its advantage of working with a potentially infinite number of features in an HDP framework, the mIBP-HDP model did not achieve a satisfactory performance in comparison with the other proposed models. However, the results were obtained by automatically selecting only 2% of distinct feature values from the entire set of values extracted from both corpora. When compared with the restricted set of features considered by the HDP$_{flat}$ and HDP$_{struct}$ models, the percentage of values selected by mIBP-HDP is only 6%. A future research area for improving this model is to consider other distributions for automatic selection of salient feature values.

**iFHMM-iHMM** In spite of the automatic feature selection employed for the iFHMM-iHMM model, its results remain competitive against the results of the HDP extensions (where the feature types were hand tuned). As shown in Table 2, most of the iFHMM-iHMM results fall in between the HDP$_{flat}$ and HDP$_{struct}$ models. Also, these results indicate that the iFHMM-iHMM model is a better framework than HDP in capturing the event mention dependencies simulated by the mIBP feature sampling scheme. Similar to the mIBP-HDP model, to achieve these results, the iFHMM-iHMM model uses only 2% values from the entire set of distinct feature values. For the experiments of the iFHMM-iHMM results reported in Table 2, we set $\alpha'$=50, $\gamma'$=0.5, and $\delta'$=0.5.

## 6 Conclusion

In this paper, we have described how a sequence of unsupervised, nonparametric Bayesian models can be employed to cluster complex linguistic objects that are characterized by a rich set of features. The experimental results proved that these models are able to solve real data applications in which the feature and cluster numbers are treated as free parameters, and the selection of features is performed automatically. While the results of event coreference resolution are promising, we believe that the classes of models proposed in this paper have a real utility for a wide range of applications.

## Footnotes

[1]For consistency, we try to preserve the notation of the original models.

[2]In this subsection and the following section, the feature term is used in context of a feature type.

[3]A0 annotates in PB a specific type of semantic role which represents the AGENT, the DOER, or the ACTOR of a specific event. Another PB argument is A1, which plays the role of the PATIENT, the THEME, or the EXPERIENCER of an event.

[4]$\mathbf{Z}^{-i,j}$ represents a notation for $\mathbf{Z} - \{Z_{i,j}\}$.

[5]In this section, a feature is represented by a (feature type:feature value) pair.

[6]Technical details for computing this probability are described in [30].

[7]The idea of using this procedure is inspired from [29] where a slice variable was used to sample a finite number of state trajectories in the iHMM.

[8]This resource is available at http://www.hlt.utdallas.edu/~ady. The annotation process is described in [7].

# References

[1] David Ahn. 2006. The stages of event extraction. In *Proceedings of the Workshop on Annotating and Reasoning about Time and Events*, pages 1–8.

[2] Amit Bagga and Breck Baldwin. 1998. Algorithms for Scoring Coreference Chains. In *Proc. of LREC*.

[3] Amit Bagga and Breck Baldwin. 1999. Cross-Document Event Coreference: Annotations, Experiments, and Observations. In *Proceedings of the ACL-99 Workshop on Coreference and its Applications*.

[4] Collin F. Baker, Charles J. Fillmore, and John B. Lowe. 1998. The Berkeley FrameNet project. In *Proceedings of COLING-ACL*.

[5] Matthew J. Beal, Zoubin Ghahramani, and Carl Edward Rasmussen. 2002. The Infinite Hidden Markov Model. In *Proceedings of NIPS*.

[6] Cosmin Adrian Bejan. 2007. Deriving Chronological Information from Texts through a Graph-based Algorithm. In *Proceedings of FLAIRS-2007*.

[7] Cosmin Adrian Bejan and Sanda Harabagiu. 2008. A Linguistic Resource for Discovering Event Structures and Resolving Event Coreference. In *Proceedings of LREC-2008*.

[8] Cosmin Adrian Bejan and Chris Hathaway. 2007. UTD-SRL: A Pipeline Architecture for Extracting Frame Semantic Structures. In *Proceedings of SemEval-2007*.

[9] Christiane Fellbaum. 1998. *WordNet: An Electronic Lexical Database*. MIT Press.

[10] Thomas S. Ferguson. 1973. A Bayesian Analysis of Some Nonparametric Problems. *The Annals of Statistics*, 1(2):209–230.

[11] Jenny Rose Finkel and Christopher D. Manning. 2008. Enforcing Transitivity in Coreference Resolution. In *Proceedings of ACL/HLT-2008*, pages 45–48.

[12] Stuart Geman and Donald Geman. 1984. Stochastic relaxation, Gibbs distributions and the Bayesian restoration of images. . *IEEE Transactions on Pattern Analysis and Machine Intelligence*, 6:721–741.

[13] Z. Ghahramani and M. Jordan. 1997. Factorial Hidden Markov Models. *Machine Learning*, 29:245–273.

[14] Zoubin Ghahramani, T. L. Griffiths, and Peter Sollich, 2007. *Bayesian Statistics 8*, chapter Bayesian nonparametric latent feature models, pages 201–225. Oxford University Press.

[15] Tom Griffiths and Zoubin Ghahramani. 2006. Infinite Latent Feature Models and the Indian Buffet Process. In *Proceedings of NIPS*, pages 475–482.

[16] Aria Haghighi and Dan Klein. 2007. Unsupervised Coreference Resolution in a Nonparametric Bayesian Model. In *Proceedings of the ACL*.

[17] Kevin Humphreys, Robert Gaizauskas, Saliha Azzam. 1997. Event Coreference for Information Extraction. In *Proceedings of the Workshop on Operational Factors in Practical, Robust Anaphora Resolution for Unrestricted Texts, 35th Meeting of ACL*, pages 75–81.

[18] LDC-ACE05. 2005. ACE (Automatic Content Extraction) English Annotation Guidelines for Events.

[19] X. Luo. 2005. On Coreference Resolution Performance Metrics. In *Proceedings of EMNLP*.

[20] X. Luo, A. Ittycheriah, H. Jing, N. Kambhatla, and S. Roukos 2004. A Mention-Synchronous Coreference Resolution Algorithm Based On the Bell Tree. In *Proceedings of ACL-2004*.

[21] Radford M. Neal. 2003. Slice Sampling. *The Annals of Statistics*, 31:705–741.

[22] Vincent Ng. 2008. Unsupervised Models for Coreference Resolution. In *Proceedings of EMNLP*.

[23] Martha Palmer, Daniel Gildea, and Paul Kingsbury. 2005. The Proposition Bank: An Annotated Corpus of Semantic Roles. *Computational Linguistics*, 31(1):71–105.

[24] Ron Papka. 1999. *On-line New Event Detection, Clustering and Tracking*. Ph.D. thesis, Department of Computer Science, University of Massachusetts.

[25] Hoifung Poon and Pedro Domingos. 2008. Joint Unsupervised Coreference Resolution with Markov Logic. In *Proceedings of EMNLP*.

[26] J. Pustejovsky, P. Hanks, R. Sauri, A. See, R. Gaizauskas, A. Setzer, D. Radev, B. Sundheim, D. Day, L. Ferro, and M. Lazo. 2003. The TimeBank Corpus. In *Corpus Linguistics*, pages 647–656.

[27] Lawrence R. Rabiner. 1989. A Tutorial on Hidden Markov Models and Selected Applications in Speech Recognition. In *Proceedings of the IEEE*, pages 257–286.

[28] Yee Whye Teh, Michael Jordan, Matthew Beal, and David Blei. 2006. Hierarchical Dirichlet Processes. *Journal of the American Statistical Association*, 101(476):1566–1581.

[29] Jurgen Van Gael, Yunus Saatci, Yee Whye Teh, and Zoubin Ghahramani. 2008. Beam Sampling for the Infinite Hidden Markov Model. In *Proceedings of ICML*, pages 1088–1095.

[30] Jurgen Van Gael, Yee Whye Teh, and Zoubin Ghahramani. 2008. The Infinite Factorial Hidden Markov Model. In *Proceedings of NIPS*.

[31] Marc Vilain, John Burger, John Aberdeen, Dennis Connolly, and Lynette Hirschman. 1995. A Model-Theoretic Coreference Scoring Scheme. In *Proceedings of MUC-6*, pages 45–52.

